# Efficient anomaly detection using bipartite $k$-NN graphs

**Kumar Sricharan**
Department of EECS
University of Michigan
Ann Arbor, MI 48104
kksreddy@umich.edu

**Alfred O. Hero III**
Department of EECS
University of Michigan
Ann Arbor, MI 48104
hero@umich.edu

## Abstract

Learning minimum volume sets of an underlying nominal distribution is a very effective approach to anomaly detection. Several approaches to learning minimum volume sets have been proposed in the literature, including the K-point nearest neighbor graph (K-kNNG) algorithm based on the geometric entropy minimization (GEM) principle [4]. The K-kNNG detector, while possessing several desirable characteristics, suffers from high computation complexity, and in [4] a simpler heuristic approximation, the leave-one-out kNNG (L1O-kNNG) was proposed. In this paper, we propose a novel bipartite $k$-nearest neighbor graph (BP-kNNG) anomaly detection scheme for estimating minimum volume sets. Our bipartite estimator retains all the desirable theoretical properties of the K-kNNG, while being computationally simpler than the K-kNNG and the surrogate L1O-kNNG detectors. We show that BP-kNNG is asymptotically consistent in recovering the p-value of each test point. Experimental results are given that illustrate the superior performance of BP-kNNG as compared to the L1O-kNNG and other state of the art anomaly detection schemes.

## 1 Introduction

Given a training set of normal events, the anomaly detection problem aims to identify unknown, anomalous events that deviate from the normal set. This novelty detection problem arises in applications where failure to detect anomalous activity could lead to catastrophic outcomes, for example, detection of faults in mission-critical systems, quality control in manufacturing and medical diagnosis.

Several approaches have been proposed for anomaly detection. One class of algorithms assumes a family of parametrically defined nominal distributions. Examples include Hotelling's T test and the Fisher F-test, which are both based on a Gaussian distribution assumption. The drawback of these algorithms is model mismatch: the supposed distribution need not be a correct representation of the nominal data, which can then lead to poor false alarm rates. More recently, several non-parametric methods based on minimum volume (MV) set estimation have been proposed. These methods aim to find the minimum volume set that recovers a certain probability mass $\alpha$ with respect to the unknown probability density of the nominal events. If a new event falls within the MV set, it is classified as normal and otherwise as anomalous.

Estimation of minimum volume sets is a difficult problem, especially for high dimensional data. There are two types of approaches to this problem: (1) transform the MV estimation problem to an equivalent density level set estimation problem, which requires estimation of the nominal density; and (2) directly identify the minimal set using function approximation and non-parametric estimation [10, 6, 9]. Both types of approaches involve explicit approximation of high dimensional quant-

ities - the multivariate density function in the first case and the boundary of the minimum volume set in the second and are therefore not easily applied to high dimensional problems.

The GEM principle developed by Hero [4] for determining MV sets circumvents the above difficulties by using the asymptotic theory of random Euclidean graphs instead of function approximation. However, the GEM based K-kNNG anomaly detection scheme proposed in [4] is computationally difficult. To address this issue, a surrogate L1O-kNNG anomaly detection scheme was proposed in [4]. L1O-kNNG is computationally simpler than K-kNNG, but loses some desirable properties of the K-kNNG, including asymptotic consistency, as shown below.

In this paper, we use the GEM principle to develop a bipartite $k$-nearest neighbor ($k$-NN) graph-based anomaly detection algorithm. BP-kNNG retains the desirable properties of the GEM principle and as a result inherits the following features: (i) it is not restricted to linear or even convex decision regions, (ii) it is completely non-parametric, (iii) it is optimal in that it converges to the uniformly most powerful (UMP) test when the anomalies are drawn from a mixture of the nominal density and the uniform density, (iv) it does not require knowledge of anomalies in the training sample, (v) it is asymptotically consistent in recovering the p-value of the test point and (vi) it produces estimated p-values, allowing for false positive rate control.

K-LPE [13] and RRS [7] are anomaly detection methods which are also based on $k$-NN graphs. BP-kNNG differs from L1O-kNNG, K-LPE and RRS in the following respects. L1O-kNNG, K-LPE and RRS do not use bipartite graphs. We will show that the bipartite nature of BP-kNNG results in significant computational savings. In addition, the K-LPE and RRS test statistics involve only the $k$-th nearest neighbor distance, while the statistic in BP-kNNG, like the L1O-kNNG, involves summation of the power weighted distance of all the edges in the $k$-NN graph. This will result in increased robustness to outliers in the training sample. Finally, we will show that the mean square rate of convergence of p-values in BP-kNNG ($O(T^{-2/(2+d)})$) is faster as compared to the convergence rate of K-LPE ($O(T^{-2/5}+T^{-6/5d})$), where $T$ is the size of the nominal training sample and $d$ is the dimension of the data.

The rest of this paper is organized as follows. In Section 2, we outline the statistical framework for minimum volume set anomaly detection. In Section 3, we describe the GEM principle and the K-kNNG and L1O-kNNG anomaly detection schemes proposed in [4]. Next, in Section 4, we develop our bipartite $k$-NN graph (BP-kNNG) method for anomaly detection. We show consistency of the method and compare its computational complexity with that of the K-kNNG, L1O-kNNG and K-LPE algorithms. In Section 5, we show simulation results that illustrate the superior performance of BP-kNNG over L1O-kNNG. We also show that our method compares favorably to other state of the art anomaly detection schemes when applied to real world data from the UCI repository [1]. We conclude with a short discussion in Section 6.

## 2  Statistical novelty detection

The problem setup is as follows. We assume that a training sample $\mathcal{X}_T = \{X_1, \ldots, X_T\}$ of $d$-dimensional vectors is available. Given a new sample $X$, the objective is to declare $X$ to either be a 'nominal' event consistent with $\mathcal{X}_T$ or an 'anomalous' event which deviates from $\mathcal{X}_T$. We seek to find a functional $D$ and corresponding detection rule $D(x) > 0$ so that $X$ is declared to be nominal if $D(x) > 0$ holds and anomalous otherwise. The acceptance region is given by $A = \{x : D(x) > 0\}$. We seek to further constrain the choice of $D$ to allow as few false negatives as possible for a fixed allowance of false positives.

To formulate this problem, we adopt the standard statistical framework for testing composite hypotheses. We assume that the training sample $\mathcal{X}_T$ is an i.i.d sample draw from an unknown $d$-dimensional probability distribution $f_0(x)$ on $[0,1]^d$. Let $X$ have density $f$ on $[0,1]^d$. The anomaly detection problem can be formulated as testing the hypotheses $H_0 : f = f_0$ versus $H_1 : f \neq f_0$.

For a given $\alpha \in (0,1)$, we seek an acceptance region $A$ that satisfies $Pr(X \in A|H_0) \geq 1 - \alpha$. This requirement maintains the false positive rate at a level no greater than $\alpha$. Let $\mathcal{A} = \{A : \int_A f_0(x)dx \geq 1 - \alpha\}$ denote the collection of acceptance regions of level $\alpha$. The most suitable acceptance region from the collection $\mathcal{A}$ would be the set which minimizes the false negative rate. Assume that the density $f$ is bounded above by some constant $C$. In this case the false negative rate is bounded by $C\lambda(A)$ where $\lambda(.)$ is the Lebesgue measure in $\mathbb{R}^d$. Consider the relaxed problem of

minimizing the upper bound $C\lambda(A)$ or equivalently the volume $\lambda(A)$ of $A$. The optimal acceptance region with a maximum false alarm rate $\alpha$ is therefore given by the minimum volume set of level $\alpha$: $\Lambda_\alpha = min\{\lambda(A) : \int_A f_0(x)dx \geq \alpha\}$.

Define the minimum entropy set of level $\alpha$ to be $\Omega_\alpha = \min\{H_\nu(A) : \int_A f_0(x)dx \geq 1 - \alpha\}$ where $H_\nu(A) = (1 - \nu)^{-1} \int_A f_0^\nu(x)dx$ is the Rényi $\nu$-entropy of the density $f_0$ over the set $A$. It can be shown that when $f_0$ is a Lebesgue density in $\mathbb{R}^d$, the minimum volume set and the minimum entropy set are equivalent, i.e. $\Lambda_\alpha$ and $\Omega_\alpha$ are identical. Therefore, the optimal decision rule for a given level of false alarm $\alpha$ is to declare an anomaly if $X \notin \Omega_\alpha$.

This decision rule has a strong optimality property [4]: when $f_0$ is Lebesgue continuous and has no 'flat' regions over its support, this decision rule is a *uniformly most powerful* (UMP) test at level $1 - \alpha$ for the null hypothesis that the test point has density $f(x)$ equal to the nominal $f_0(x)$ versus the alternative hypothesis that $f(x) = (1 - \epsilon)f_0(x) + \epsilon U(x)$, where $U(x)$ is the uniform density over $[0, 1]^d$ and $\epsilon \in [0, 1]$. Furthermore, the power function is given by $\beta = Pr(X \notin \Omega_\alpha|H_1) = (1 - \epsilon)\alpha + \epsilon(1 - \lambda(\Omega_\alpha))$.

## 3    GEM principle

In this section, we briefly review the geometric entropy minimization (GEM) principle method [4] for determining minimum entropy sets $\Omega_\alpha$ of level $\alpha$. The GEM method directly estimates the critical region $\Omega_\alpha$ for detecting anomalies using minimum coverings of subsets of points in a nominal training sample. These coverings are obtained by constructing minimal graphs, e.g., the $k$-minimal spanning tree or the $k$-nearest neighbor graph, covering a $K$-point subset that is a given proportion of the training sample. Points in the training sample that are not covered by the K-point minimal graphs are identified as tail events.

In particular, let $\mathcal{X}_{K,T}$ denote one of the $\binom{T}{K}$ $K$ point subsets of $\mathcal{X}_T$. The $k$-nearest neighbors ($k$-NN) of a point $X_i \in \mathcal{X}_{K,T}$ are the $k$ closest points to $X_i$ among $\mathcal{X}_{K,T} - X_i$. Denote the corresponding set of edges between $X_i$ and its $k$-NN by $\{e_{i(1)}, \ldots, e_{i(k)}\}$. For any subset $\mathcal{X}_{K,T}$, define the total power weighted edge length of the $k$-NN graph on $\mathcal{X}_{K,T}$ with power weighting $\gamma$ ($0 < \gamma < d$), as

$$L_{kNN}(\mathcal{X}_{K,T}) = \sum_{i=1}^{K} \sum_{l=1}^{k} |e_{t_i(l)}|^\gamma,$$

where $\{t_1, \ldots, t_K\}$ are the indices of $X_i \in \mathcal{X}_{K,T}$. Define the K-kNNG graph to be the K-point $k$-NN graph having minimal length $\min_{\mathcal{X}_{T,K} \in \mathcal{X}_T} L_{kNN}(\mathcal{X}_{T,K})$ over all $\binom{T}{K}$ subsets $\mathcal{X}_{K,T}$. Denote the corresponding length minimizing subset of $K$ points by $\mathcal{X}_{T,K}^* = \underset{\mathcal{X}_{T,K} \in \mathcal{X}}{\operatorname{argmin}} L_{kNN}(\mathcal{X}_{K,T})$.

The K-kNNG thus specifies a minimal graph covering $\mathcal{X}_{K,T}^*$ of size $K$. This graph can be viewed as capturing the densest regions of $\mathcal{X}_T$. If $\mathcal{X}_T$ is an i.i.d. sample from a multivariate density $f_0(x)$ and if $\lim_{K,T \to \infty} K/T = \rho$, then the set $\mathcal{X}_{K,T}^*$ converges a.s. to the minimum $\nu$-entropy set containing a proportion of at least $\rho$ of the mass of $f_0(x)$, where $\nu = 1 - \gamma/d$ [4]. This set can be used to perform anomaly detection.

### 3.1    K-kNNG anomaly detection

Given a test sample $X$, denote the pooled sample $\mathcal{X}_{T+1} = \mathcal{X}_T \cup \{X\}$ and determine the K-kNNG graph over $\mathcal{X}_{T+1}$. Declare $X$ to be an anomaly if $X \notin \mathcal{X}_{K,T+1}^*$ and nominal otherwise. When the density $f_0$ is Lebesgue continuous, it follows from [4] that as $K, T \to \infty$, this anomaly detection algorithm has false alarm rate that converges to $\alpha = 1 - K/T$ and power that converges to that of the minimum volume set test of level $\alpha$. An identical detection scheme based on the $K$-minimal spanning tree has also been developed in [4].

The K-kNNG anomaly detection scheme therefore offers a direct approach to detecting outliers while bypassing the more difficult problems of density estimation and level set estimation in high dimensions. However, this algorithm requires construction of k-nearest neighbor graphs (or k-minimal spanning trees) over $\binom{T}{K}$ different subsets. For each input test point, the runtime of this algorithm

is therefore $O(dK^2\binom{T}{K})$. As a result, the K-kNNG method is not well suited for anomaly detection for large sample sizes.

## 3.2 L1O-kNNG

To address the computational problems of K-kNNG, Hero [4] proposed implementing the K-kNNG for the simplest case $K = T - 1$. The runtime of this algorithm for each input test point is $O(dT^2)$. Clearly, the L1O-kNNG is of much lower complexity that the K-kNNG scheme. However, the L1O-kNNG detects anomalies at a fixed false alarm rate $1/(T + 1)$, where $T$ is the training sample size. To detect anomalies at a higher false alarm rate $\alpha^*$, one would have to subsample the training set and only use $T^* = 1/\alpha^* - 1$ training samples. This destroys any hope for asymptotic consistency of the L1O-kNNG.

In the next section, we propose a different GEM based algorithm that uses bipartite graphs. The algorithm has algorithm has a much faster runtime than the L1O-kNNG, and unlike the L1O-kNNG, is asymptotically consistent and can operate at any specified alarm rate $\alpha$. We describe our algorithm below.

## 4 BP-kNNG

Let $\{\mathcal{X}_N, \mathcal{X}_M\}$ be a partition of $\mathcal{X}_T$ with $card\{\mathcal{X}_N\} = N$ and $card\{\mathcal{X}_M\} = M = T - N$ respectively. As above, let $\mathcal{X}_{K,N}$ denote one of the $\binom{N}{K}$ subsets of $K$ distinct points from $\mathcal{X}_N$. Define the bipartite $k$-NN graph on $\{\mathcal{X}_{K,N}, \mathcal{X}_M\}$ to be the set of edges linking each $X_i \in \mathcal{X}_{K,N}$ to its $k$ nearest neighbors in $\mathcal{X}_M$. Define the total power weighted edge length of this bipartite $k$-NN graph with power weighting $\gamma$ ($0 < \gamma < d$) and a fixed number of edges $s$ ($1 \leq s \leq k$) corresponding to each vertex $X_i \in \mathcal{X}_{K,N}$ to be

$$L_{s,k}(\mathcal{X}_{K,N}, \mathcal{X}_M) = \sum_{i=1}^{K} \sum_{l=k-s+1}^{k} |e_{t_i(l)}|^\gamma,$$

where $\{t_1, \ldots, t_K\}$ are the indices of $X_i \in \mathcal{X}_{K,N}$ and $\{e_{t_i(1)}, \ldots, e_{t_i(k)}\}$ are the $k$-NN edges in the bipartite graph originating from $X_{t_i} \in \mathcal{X}_{K,N}$. Define the bipartite K-kNNG graph to be the one having minimal weighted length $\min_{\mathcal{X}_{N,K} \in \mathcal{X}_N} L_{s,k}(\mathcal{X}_{N,K}, \mathcal{X}_M)$ over all $\binom{N}{K}$ subsets $\mathcal{X}_{K,N}$. Define the corresponding minimizing subset of $K$ points of $\mathcal{X}_{K,N}$ by $\mathcal{X}^*_{K,N} = \underset{\mathcal{X}_{K,N} \in \mathcal{X}}{\operatorname{argmin}} L_{s,k}(\mathcal{X}_{K,N}, \mathcal{X}_M)$.

Using the theory of partitioned $k$-NN graph entropy estimators [11], it follows that as $k/M \to 0$, $k, N \to \infty$ and for fixed $s$, the set $\mathcal{X}^*_{K,N}$ converges a.s. to the minimum $\nu$-entropy set $\Omega_{1-\rho}$ containing a proportion of at least $\rho$ of the mass of $f_0(x)$, where $\rho = \lim_{K,N \to \infty} K/N$ and $\nu = 1 - \gamma/d$.

This suggests using the bipartite $k$-NN graph to detect anomalies in the following way. Given a test point $X$, denote the pooled sample $\mathcal{X}_{N+1} = \mathcal{X}_N \cup \{X\}$ and determine the optimal bipartite K-kNNG graph $\mathcal{X}^*_{K,N+1}$ over $\{\mathcal{X}_{K,N+1}, \mathcal{X}_M\}$. Now declare $X$ to be an anomaly if $X \notin \mathcal{X}^*_{K,N+1}$ and nominal otherwise. It is clear that by the GEM principle, this algorithm detects false alarms at a rate that converges to $\alpha = 1 - K/T$ and power that converges to that of the minimum volume set test of level $\alpha$.

We can equivalently determine $\mathcal{X}^*_{K,N+1}$ as follows. For each $X_i \in \mathcal{X}_N$, construct $d_{s,k}(X_i) = \sum_{l=k-s+1}^{k} |e_{i(l)}|^\gamma$. For each test point $X$, define $d_{s,k}(X) = \sum_{l=s-k+1}^{k} |e_{X(l)}|^\gamma$, where $\{e_{X(1)}, \ldots, e_{X(k)}\}$ are the $k$-NN edges from $X$ to $\mathcal{X}_M$. Now, choose the $K$ points among $\mathcal{X}_N \cup X$ with the $K$ smallest of the $N + 1$ edge lengths $\{d_{s,k}(X_i), X_i \in \mathcal{X}_N\} \cup \{d_{s,k}(X)\}$. Because of the bipartite nature of the construction, this is equivalent to choosing $\mathcal{X}^*_{K,N+1}$. This leads to the proposed BP-kNNG anomaly detection algorithm described by Algorithm 1.

### 4.1 BP-kNNG p-value estimates

The p-value is a score between 0 and 1 that is associated with the likelihood that a given point $X_0$ comes from a specified nominal distribution. The BP-kNNG generates an estimate of the p-value

---
**Algorithm 1** Anomaly detection scheme using bipartite $k$-NN graphs
---
1. Input: Training samples $\mathcal{X}_T$, test samples $X$, false alarm rate $\alpha$
2. Training phase
a. Create partition $\{\mathcal{X}_N, \mathcal{X}_M\}$
b. Construct $k$-NN bipartite graph on partition
c. Compute $k$-NN lengths $d_{s,k}(X_i)$ for each $X_i \in \mathcal{X}_N$: $d_{s,k}(X_i) = \sum_{l=k-s+1}^{k} |e_{i(l)}|^\gamma$
3. Test phase: detect anomalous points
**for** each input test sample $X$ **do**
    Compute $k$-NN length $d_{s,k}(X) = \sum_{l=k-s+1}^{k} |e_{X(l)}|^\gamma$
    **if**
$$(1/N) \sum_{X_i \in \mathcal{X}_N} 1(d_{s,k}(X_i) < d_{s,k}(X)) \geq 1 - \alpha$$
   **then**
     Declare $X$ to be anomalous
   **else**
     Declare $X$ to be non-anomalous
   **end if**
**end for**
---

that is asymptotically consistent, guaranteeing that the BP-kNNG detector is a consistent novelty detector.

Specifically, for a given test point $X_0$, the true p-value associated with a point $X_0$ in a minimum volume set test is given by $p_{true}(X_0) = \int_{S(X_0)} f_0(z)dz$ where $S(X_0) = \{z : f_0(z) \leq f_0(X_0)\}$ and $E(X_0) = \{z : f_0(z) = f_0(X_0)\}$. $p_{true}(X_0)$ is the minimal level $\alpha$ at which $X_0$ would be rejected. The empirical p-value associated with the BP-kNNG is defined as

$$p_{bp}(X_0) \quad = \quad \frac{\sum_{X_i \in \mathcal{X}_N} 1(d_{s,k}(X_i) \geq d_{s,k}(X_0))}{N}. \tag{1}$$

## 4.2 Asymptotic consistency and optimal convergence rates

Here we prove that the BP-kNNG detector is asymptotically consistent by showing that for a fixed number of edges $s$, $\mathbb{E}[(p_{bp}(X_0) - p_{true}(X_0))^2] \to 0$ as $k/M \to 0$, $k, N \to \infty$. In the process, we also obtain rates of convergence of this mean-squared error. These rates depend on $k$, $N$ and $M$ and result in the specification of an optimal number of neighbors $k$ and an optimal partition ratio $N/M$ that achieve the best trade-off between bias and variance of the p-value estimates $p_{bp}(X_0)$. We assume that the density $f_0$ (i) is bounded away from $0$ and $\infty$ and is continuous on its support $\mathcal{S}$, (ii) has no flat spots over its support set and (iii) has a finite number of modes. Let $\mathbb{E}$ denote the expectation w.r.t. the density $f_0$, and $\mathbb{B}$, $\mathbb{V}$ denote the bias and variance operators. Throughout this section, assume without loss of generality that $\{X_1, \ldots, X_N\} \in \mathcal{X}_N$ and $\{X_{N+1}, \ldots, X_T\} \in \mathcal{X}_M$.

**Bias:** We first introduce the oracle p-value $p_{orac}(X_0) = (1/N)\sum_{X_i \in \mathcal{X}_N} 1(f_0(X_i) \leq f_0(X_0))$ and note that $\mathbb{E}[p_{orac}(X_0)] = p_{true}(X_0)$. The distance $e_{i(l)}$ of a point $X_i \in \mathcal{X}_N$ to its $l$-th nearest neighbor in $\mathcal{X}_M$ is related to the bipartite $l$-nearest neighbor density estimate $\hat{f}_l(X_i) = (l-1)/(Mc_d e_{i(l)}^d)$ (section 2.3, [11]) where $c_d$ is the unit ball volume in $d$ dimensions. Let

$$e(X) = \left( \sum_{l=k-s+1}^{k} \left( \frac{k-1}{l-1} \hat{f}_l(X) \right)^{\nu-1} \right) - s(f(X))^{\nu-1}$$

and

$$\delta(X_i, X_0) = \delta_i = (f(X_i))^{\nu-1} - (f(X_0))^{\nu-1}.$$

We then have

$$\begin{aligned} \mathbb{B}[p_{bp}(X_0)] \quad &= \quad \mathbb{E}[p_{bp}(X_0)] - p_{true}(X_0) = \mathbb{E}[p_{bp}(X_0) - p_{orac}(X_0)] \\ &= \quad \mathbb{E}[1(d_{s,k}(X_1) \geq d_{s,k}(X_0))] - \mathbb{E}[1(f(X_1) \leq f(X_0))] \\ &= \quad \mathbb{E}[1(e(X_1) - e(X_0) + \delta_1 \leq 0) - 1(\delta_1 \leq 0)]. \end{aligned}$$

This bias will be non-zero when $1(e(X_1) - e(X_0) + \delta_1 \leq 0) \neq 1(\delta_1 \leq 0)$. First we investigate this condition when $\delta_1 > 0$. In this case, for $1(e(X_1) - e(X_0) + \delta_1 \leq 0) \neq 1(\delta_1 \leq 0)$, we need $-e(X_1) + e(X_0) \geq \delta_1$. Likewise, when $\delta_1 \leq 0$, $1(e(X_1) - e(X_0) + \delta_1 \leq 0) \neq 1(\delta_1 \leq 0)$ occurs when $e(X_1) - e(X_0) > |\delta_1|$.

From the theory developed in [11], for any fixed $s$, $|e(X)| = O(k/M)^{1/d} + O(1/\sqrt{k})$ with probability greater than $1 - o(1/M)$. This implies that

$$
\begin{aligned}
\mathbb{B}[p_{bp}(X_0)] &= \mathbb{E}[1(e(X_1) - e(X_0) + \delta_1 \leq 0) - 1(\delta_1 \leq 0)] \\
&= Pr\{|\delta_1| = O((k/M)^{1/d} + 1/\sqrt{k})\} + o(1/M) = O((k/M)^{1/d} + 1/\sqrt{k}), \quad (2)
\end{aligned}
$$

where the last step follows from our assumption that the density $f_0$ is continuous and has a finite number of modes.

**Variance:** Define $b_i = 1(e(X_i) - e(X_0) + \delta_i \leq 0) - 1(\delta_i \leq 0)$. We can compute the variance in a similar manner to the bias as follows (for additional details, please refer to the supplementary material):

$$
\begin{aligned}
\mathbb{V}[p_{bp}(X_0)] &= \frac{1}{N}\mathbb{V}[1(e(X_1) - e(X_0) + \delta_1 \leq 0)] + \frac{N-1}{N}Cov[b_1, b_2] \\
&= O(1/N) + \mathbb{E}[b_1 b_2] - (\mathbb{E}[b_1]\mathbb{E}[b_2]) = O(1/N + (k/M)^{2/d} + 1/k). \quad (3)
\end{aligned}
$$

**Consistency of p-values:** From (2) and (3), we obtain an asymptotic representation of the estimated p-value $\mathbb{E}[(p_{bp}(X_0) - p_{true}(X_0))^2] = O((k/M)^{2/d}) + O(1/k) + O(1/N)$. This implies that $p_{bp}$ converges in mean square to $p_{true}$, for a fixed number of edges $s$, as $k/M \to 0$, $k, N \to \infty$.

**Optimal choice of parameters:** The optimal choice of $k$ to minimize the MSE is given by $k = \Theta(M^{2/(2+d)})$. For fixed $M + N = T$, to minimize MSE, $N$ should then be chosen to be of the order $O(M^{(4+d)/(4+2d)})$, which implies that $M = \Theta(T)$. The mean square convergence rate for this optimal choice of $k$ and partition ratio $N/M$ is given by $O(T^{-2/(2+d)})$. In comparison, the K-LPE method requires that $k$ grows with the sample size at rate $k = \Theta(T^{2/5})$. The mean square rate of convergence of the p-values in K-LPE is then given by $O(T^{-2/5} + T^{-6/5d})$. The rate of convergence of the p-values is therefore faster in the case of BP-kNNG as compared to K-LPE.

### 4.3 Comparison of run time complexity

Here we compare complexity of BP-kNNG with that of K-kNNG, L1O-kNNG and K-LPE. For a single query point $X$, the runtime of K-kNNG is $O(dK^2\binom{T}{K})$, while the complexity of the surrogate L1O-kNN algorithm and the K-LPE is $O(dT^2)$. On the other hand, the complexity of the proposed BP-kNNG algorithm is dominated by the computation of $d_k(X_i)$ for each $X_i \in \mathcal{X}_N$ and $d_k(X)$, which is $O(dNM) = O(dT^{(8+3d)/(4+2d)}) = o(dT^2)$.

For the K-kNNG, L1O-kNNG and K-LPE, a new $k$-NN graph has to be constructed on $\{\mathcal{X}_N \cup \{X\}\}$ for every new query point $X$. On the other hand, because of the bipartite construction of our $k$-NN graph, $d_k(X_i)$ for each $X_i \in \mathcal{X}_N$ needs to be computed and stored only once. For every new query $X$ that comes in, the cost to compute $d_k(X)$ is only $O(dM) = O(dT)$. For a total of $L$ query points, the overall runtime complexity of our algorithm is therefore much smaller than the L1O-kNNG, K-LPE and K-kNNG anomaly detection schemes ($O(dT(T^{(4+d)/(4+2d)} + L))$ compared to $O(dLT^2)$, $O(dLT^2)$ and $O(dLK^2\binom{T}{K})$) respectively).

## 5 Simulation comparisons

We compare the L1O-kNNG and the bipartite K-kNNG schemes on a simulated data set. The training set contains 1000 realizations drawn from a 2-dimensional Gaussian density $f_0$ with mean 0 and diagonal covariance with identical component variances of $\sigma = 0.1$. The test set contains 500 realizations drawn from $0.8f_0 + 0.2U$, where $U$ is the uniform density on $[0, 1]^2$. Samples from the uniform distribution are classified to be anomalies. The percentage of anomalies in the test set is therefore 20%.

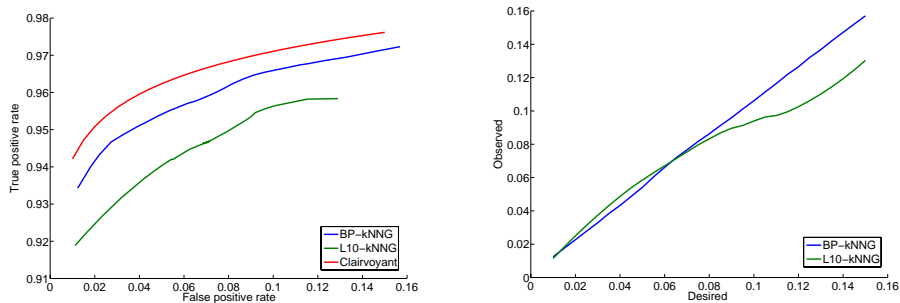

(a) ROC curves for L1O-kNNG and BP-kNNG. (b) Comparison of observed false alarm rates for The labeled 'clairvoyant' curve is the ROC of the L1O-kNNG and BP-kNNG with the desired false UMP anomaly detector.. alarm rates.

Figure 1: Comparison of performance of L1O-kNNG and BP-kNNG.

| Data set | Sample size | Dimension | Anomaly class |
|---|---|---|---|
| HTTP (KDD'99) | 567497 | 3 | attack (0.4%) |
| Forest | 286048 | 10 | class 4 vs class 2 (0.9%) |
| Mulcross | 262144 | 4 | 2 clusters (10%) |
| SMTP (KDD'99) | 95156 | 3 | attack (0.03%) |
| Shuttle | 49097 | 9 | class 2,3,5,6,7 vs class 1 (7%) |

Table 1: Description of data used in anomaly detection experiments.

The distribution $f_0$ has essential support on the unit square. For this simple case the minimum volume set of level $\alpha$ is a disk centered at the origin with radius $\sqrt{2\sigma^2 \log(1/\alpha)}$. The power of the uniformly most powerful (UMP) test is $1 - 2\pi\sigma^2 \log(1/\alpha)$.

L1O-kNNG and BP-kNNG were implemented in Matlab 7.6 on an 2 GHz Intel processor with 3 GB of RAM. The value of $k$ was set to 5. For the BP-kNNG, we set $s = 1$, $N = 100$ and $M = 900$. In Fig. 1(a), we compare the detection performance of L1O-kNNG and BP-kNNG against the 'clairvoyant' UMP detector in terms of the ROC. We note that the proposed BP-kNNG is closer to the optimal UMP test as compared to the L1O-kNNG. In Fig. 1(b) we note the close agreement between desired and observed false alarm rates for BP-kNNG. Note that the L1O-kNNG significantly underestimates its false alarm rate for higher levels of true false alarm. In the case of the L1O-kNNG, it took an average of 60ms to test each instance for possible anomaly. The total run-time was therefore 60x500 = 3000ms. For the BP-kNNG, for a single instance, it took an average of 57ms. When all the instances were processed together, the total run time was only 97ms. This significant savings in runtime is due to the fact that the bipartite graph does not have to be constructed separately for each new test instance; it suffices to construct it once on the entire data set.

## 5.1 Experimental comparisons

In this section, we compare our algorithm to several other state of the art anomaly detection algorithms, namely: MassAD [12], isolation forest (or iForest) [5], two distance-based methods ORCA [2] and K-LPE [13], a density-based method LOF [3], and the one-class support vector machine (or 1-SVM) [9]. All the methods are tested on the five largest data sets used in [5]. The data characteristics are summarized in Table 1. One of the anomaly data generators is Mulcross [8] and the other four are from the UCI repository [1]. Full details about the data can be found in [5].

The comparison performance is evaluated in terms of averaged AUC (area under ROC curve) and processing time (a total of training and test time). Results for BP-kNNG are compared with results for L1O-kNNG, K-LPE, MassAD, iForest and ORCA in Table 2. The results for MassAD, iForest and ORCA are reproduced from [12]. MassAD and iForest were implemented in Matlab and tested on an AMD Opteron machine with a 1.8 GHz processor and 4 GB memory. The results for ORCA,

| Data sets | AUC | | | | | | Time (secs) | | | | | |
|---|---|---|---|---|---|---|---|---|---|---|---|---|
| | BP | L10 | K-LPE | Mass | iF | ORCA | BP | L10 | K-LPE | Mass | iF | ORCA |
| HTTP | 0.99 | NA | NA | 1.00 | 1.00 | 0.36 | 3.81 | .10/i | .19/i | 34 | 147 | 9487 |
| Forest | 0.86 | NA | NA | 0.91 | 0.87 | 0.83 | 7.54 | .18/i | .18/i | 18 | 79 | 6995 |
| Mulcross | 1.00 | NA | NA | 0.99 | 0.96 | 0.33 | 4.68 | .26/i | .17/i | 17 | 75 | 2512 |
| SMTP | 0.90 | NA | NA | 0.86 | 0.88 | 0.87 | 0.74 | .11/i | .17/i | 7 | 26 | 267 |
| Shuttle | 0.99 | NA | NA | 0.99 | 1.00 | 0.60 | 1.54 | .45/i | .16/i | 4 | 15 | 157 |

Table 2: Comparison of anomaly detection schemes in terms of AUC and run-time for BP-kNNG (BP) against L1O-kNNG (L10), K-LPE, MassAD (Mass), iForest (iF) and ORCA. When reporting results for L1O-kNNG and K-LPE, we report the processing time per test instance (/i). We are unable to report the AUC for K-LPE and L1O-kNNG because of the large processing time. We note that BP-kNNG compares favorably in terms of AUC while also requiring the least run-time.

| Data sets | Desired false alarm | | | | |
|---|---|---|---|---|---|
| | 0.01 | 0.02 | 0.05 | 0.1 | 0.2 |
| HTTP (KDD'99) | 0.007 | 0.015 | 0.063 | 0.136 | 0.216 |
| Forest | 0.009 | 0.015 | 0.035 | 0.071 | 0.150 |
| Mulcross | 0.008 | 0.014 | 0.040 | 0.096 | 0.186 |
| SMTP (KDD'99) | 0.006 | 0.017 | 0.046 | 0.099 | 0.204 |
| Shuttle | 0.026 | 0.030 | 0.045 | 0.079 | 0.179 |

Table 3: Comparison of desired and observed false alarm rates for BP-kNNG. There is good agreement between the desired and observed rates.

LOF and 1-SVM were conducted using the same experimental setting but on a faster 2.3 GHz machine. We exclude the results for LOF and 1-SVM in table 2 because MassAD, iForest and ORCA have been shown to outperform LOF and 1-SVM in [12].

We implemented BP-kNNG, L1O-kNNG and K-LPE in Matlab on an Intel 2 GHz processor with 3 GB RAM. We note that this machine is comparable to the AMD Opteron machine with a 1.8 GHz processor. We choose $T = 10^4$ training samples and fix $k = 50$ in all three cases. For BP-kNNG, we fix $s = 5$ and $N = 10^3$. When reporting results for L1O-kNNG and K-LPE, we report the processing time per test instance (/i). We are unable to report the AUC for K-LPE because of the large processing time and for L1O-kNNG because it cannot operate at high false alarm rates.

From the results in Table 2, we see that BP-kNNG performs comparably in terms of AUC to the other algorithms, while having the least processing time across all algorithms (implemented on different, but comparable machines). In addition, BP-kNNG allows the specification of a threshold for anomaly detection at a desired false alarm rate. This is corroborated by the results in Table 3, where we see that the observed false alarm rates across the different data sets are close to the desired false alarm rate.

# 6 Conclusions

The geometric entropy minimization (GEM) principle was introduced in [4] to extract minimal set coverings that can be used to detect anomalies from a set of training samples. In this paper we propose a bipartite $k$-nearest neighbor graph (BP-kNNG) anomaly detection algorithm based on the GEM principle. BP-kNNG inherits the theoretical optimality properties of GEM methods including consistency, while being an order of magnitude faster than the methods proposed in [4].

We compared BP-kNNG against state of the art anomaly detection algorithms and showed that BP-kNNG compares favorably in terms of both ROC performance and computation time. In addition, BP-kNNG enjoys several other advantages including the ability to detect anomalies at a desired false alarm rate. In BP-kNNG, the p-values of each test point can also be easily computed (1), making BP-kNNG easily extendable to incorporating false discovery rate constraints.

# References

[1] A. Asuncion and D.J. Newman. UCI machine learning repository, 2007.

[2] S. D. Bay and M. Schwabacher. Mining distance-based outliers in near linear time with randomization and a simple pruning rule. In *Proceedings of the ninth ACM SIGKDD international conference on Knowledge discovery and data mining*, KDD '03, pages 29–38, New York, NY, USA, 2003. ACM.

[3] M. M. Breunig, H. Kriegel, R. T. Ng, and J. Sander. Lof: identifying density-based local outliers. In *Proceedings of the 2000 ACM SIGMOD international conference on Management of data*, SIGMOD '00, pages 93–104, New York, NY, USA, 2000. ACM.

[4] A. O. Hero. Geometric entropy minimization (gem) for anomaly detection and localization. In *Proc. Advances in Neural Information Processing Systems (NIPS*, pages 585–592. MIT Press, 2006.

[5] F. T. Liu, K. M. Ting, and Z. Zhou. Isolation forest. In *Proceedings of the 2008 Eighth IEEE International Conference on Data Mining*, pages 413–422, Washington, DC, USA, 2008. IEEE Computer Society.

[6] C. Park, J. Z. Huang, and Y. Ding. A computable plug-in estimator of minimum volume sets for novelty detection. *Operations Research*, 58(5):1469–1480, 2010.

[7] S. Ramaswamy, R. Rastogi, and K. Shim. Efficient algorithms for mining outliers from large data sets. *SIGMOD Rec.*, 29:427–438, May 2000.

[8] D. M. Rocke and D. L. Woodruff. Identification of Outliers in Multivariate Data. *Journal of the American Statistical Association*, 91(435):1047–1061, 1996.

[9] B. Schölkopf, R. Williamson, A. Smola, J. Shawe-Taylor, and J.Platt. Support Vector Method for Novelty Detection. volume 12, 2000.

[10] C. Scott and R. Nowak. Learning minimum volume sets. *J. Machine Learning Res*, 7:665–704, 2006.

[11] K. Sricharan, R. Raich, and A. O. Hero. Empirical estimation of entropy functionals with confidence. *ArXiv e-prints*, December 2010.

[12] K. M. Ting, G. Zhou, T. F. Liu, and J. S. C. Tan. Mass estimation and its applications. In *Proceedings of the 16th ACM SIGKDD international conference on Knowledge discovery and data mining*, KDD '10, pages 989–998, New York, NY, USA, 2010. ACM.

[13] M. Zhao and V. Saligrama. Anomaly detection with score functions based on nearest neighbor graphs. *Computing Research Repository*, abs/0910.5461, 2009.

